# Using Local Models to Control Movement

**Christopher G. Atkeson**
Department of Brain and Cognitive Sciences
and the Artificial Intelligence Laboratory
Massachusetts Institute of Technology
NE43-771, 545 Technology Square
Cambridge, MA 02139
cga@ai.mit.edu

## ABSTRACT

This paper explores the use of a model neural network for motor learning. Steinbuch and Taylor presented neural network designs to do nearest neighbor lookup in the early 1960s. In this paper their nearest neighbor network is augmented with a local model network, which fits a local model to a set of nearest neighbors. The network design is equivalent to local regression. This network architecture can represent smooth nonlinear functions, yet has simple training rules with a single global optimum. The network has been used for motor learning of a simulated arm and a simulated running machine.

## 1    INTRODUCTION

A common problem in motor learning is approximating a continuous function from samples of the function's inputs and outputs. This paper explores a neural network architecture that simply remembers experiences (samples) and builds a local model to answer any particular query (an input for which the function's output is desired). This network design can represent smooth nonlinear functions, yet has simple training rules with a single global optimum for building a local model in response to a query. Our approach is to model complex continuous functions using simple local models. This approach avoids the difficult problem of finding an appropriate structure for a global model. A key idea is to form a training set for the local model network after a query to be answered is known. This approach

allows us to include in the training set only relevant experiences (nearby samples). The local model network, which may be a simple network architecture such as a perceptron, forms a model of the portion of the function near the query point. This local model is then used to predict the output of the function, given the input. The local model network is retrained with a new training set to answer the next query. This approach minimizes interference between old and new data, and allows the range of generalization to depend on the density of the samples.

Steinbuch (Steinbuch 1961, Steinbuch and Piske 1963) and Taylor (Taylor 1959, Taylor 1960) independently proposed neural network designs that used a local representation to do nearest neighbor lookup and pointed out that this approach could be used for control. They used a layer of hidden units to compute an inner product of each stored vector with the input vector. A winner-take-all circuit then selected the hidden unit with the highest activation. This type of network can find nearest neighbors or best matches using a Euclidean distance metric (Kazmierczak and Steinbuch 1963). In this paper their nearest neighbor lookup network (which I will refer to as the memory network) is augmented with a local model network, which fits a local model to a set of nearest neighbors.

The ideas behind the network design used in this paper have a long history. Approaches which represent previous experiences directly and use a similar experience or similar experiences to form a local model are often referred to as nearest neighbor or k-nearest neighbor approaches. Local models (often polynomials) have been used for many years to smooth time series (Sheppard 1912, Sherriff 1920, Whittaker and Robinson 1924, Macauley 1931) and interpolate and extrapolate from limited data. Lancaster and Šalkauskas (1986) refer to nearest neighbor approaches as "moving least squares" and survey their use in fitting surfaces to arbitrarily spaced points. Eubank (1988) surveys the use of nearest neighbor estimators in nonparametric regression. Farmer and Sidorowich (1988) survey the use of nearest neighbor and local model approaches in modeling chaotic dynamic systems.

Crain and Bhattacharyya (1967), Falconer (1971), and McLain (1974) suggested using a weighted regression to fit a local polynomial model at each point a function evaluation was desired. All of the available data points were used. Each data point was weighted by a function of its distance to the desired point in the regression. McIntyre, Pollard, and Smith (1968), Pelto, Elkins, and Boyd (1968), Legg and Brent (1969), Palmer (1969), Walters (1969), Lodwick and Whittle (1970), Stone (1975) and Franke and Nielson (1980) suggested fitting a polynomial surface to a set of nearest neighbors, also using distance weighted regression. Cleveland (1979) proposed using robust regression procedures to eliminate outlying or erroneous points in the regression process. A program implementing a refined version of this approach (LOESS) is available by sending electronic mail containing the single line, *send dloess from a*, to the address netlib@research.att.com (Grosse 1989). Cleveland, Devlin and Grosse (1988) analyze the statistical properties of the LOESS algorithm and Cleveland and Devlin (1988) show examples of its use. Stone (1977, 1982), Devroye (1981), Cheng (1984), Li (1984), Farwig (1987), and Müller (1987)

provide analyses of nearest neighbor approaches. Franke (1982) compares the performance of nearest neighbor approaches with other methods for fitting surfaces to data.

## 2    THE NETWORK ARCHITECTURE

The memory network of Steinbuch and Taylor is used to find the nearest stored vectors to the current input vector. The memory network computes a measure of the distance between each stored vector and the input vector in parallel, and then a "winner take all" network selects the nearest vector (nearest neighbor). Euclidean distance has been chosen as the distance metric, because the Euclidean distance is invariant under rotation of the coordinates used to represent the input vector.

The memory network consists of three layers of units: input units, hidden or memory units, and output units. The squared Euclidean distance between the input vector ($\mathbf{i}$) and a weight vector ($\mathbf{w}_k$) for the connections of the input units to hidden unit $k$ is given by:

$$d_k^2 = (\mathbf{i} - \mathbf{w}_k)^{\mathrm{T}}(\mathbf{i} - \mathbf{w}_k) = \mathbf{i}^{\mathrm{T}}\mathbf{i} - 2\mathbf{i}^{\mathrm{T}}\mathbf{w}_k + \mathbf{w}_k^{\mathrm{T}}\mathbf{w}_k$$

Since the quantity $\mathbf{i}^{\mathrm{T}}\mathbf{i}$ is the same for all hidden units, minimizing the distance between the input vector and the weight vector for each hidden unit is equivalent to maximizing:

$$\mathbf{i}^{\mathrm{T}}\mathbf{w}_k - 1/2\mathbf{w}_k^{\mathrm{T}}\mathbf{w}_k$$

This quantity is the inner product of the input vector and the weight vector for hidden unit $k$, biased by half the squared length of the weight vector.

Dynamics of the memory network neurons allow the memory network to output a sequence of nearest neighbors. These nearest neighbors form the selected training sequence for the local model network. Memory unit dynamics can be used to allocate "free" memory units to new experiences, and to forget old training points when the capacity of the memory network is fully utilized.

The local model network consists of only one layer of modifiable weights preceded by any number of layers with fixed connections. There may be arbitrary preprocessing of the inputs of the local model, but the local model is linear in the parameters used to form the fit. The local model network using the LMS training algorithm performs a linear regression of the transformed inputs against the desired outputs. Thus, the local model network can be used to fit a linear regression model to the selected training set. With multiplicative interactions between inputs the local model network can be used to fit a polynomial surface (such as a quadratic) to the selected training set. An alternative implementation of the local model network could use a single layer of "sigma-pi" units.

This network design has simple training rules. In the memory network the weights are simply the values of the components of input and output vectors, and the bias for each memory unit is just half the squared length of the corresponding input weight vector. No search for weights is necessary, since the weights are directly

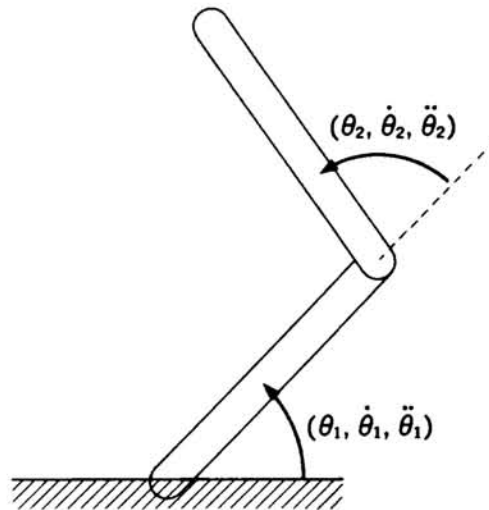

**Figure 1:** Simulated Planar Two-joint Arm

given by the data to be stored. The local model network is linear in the weights, leading to a single optimum which can be found by linear regression or gradient descent. Thus, convergence to the global optimum is guaranteed when forming a local model to answer a particular query.

This network architecture was simulated using k-d tree data structures (Friedman, Bentley, and Finkel 1977) on a standard serial computer and also using parallel search on a massively parallel computer, the Connection Machine (Hillis 1985). A special purpose computer is being built to implement this network in real time.

## 3  APPLICATIONS

The network has been used for motor learning of a simulated arm and a simulated running machine. The network performed surprisingly well in these simple evaluations. The simulated arm was able to follow a desired trajectory after only a few practice movements. Performance of the simulated running machine in following a series of desired velocities was also improved. This paper will report only on the arm trajectory learning.

Figure 1 shows the simulated 2-joint planar arm. The problem faced in this simulation is to learn the correct joint torques to drive the arm along the desired trajectory (the inverse dynamics problem). In addition to the feedforward control signal produced by the network described in this paper, a feedback controller was also used.

Figure 2 shows several learning curves for this problem. The first point in each of the curves shows the performance generated by the feedback controller alone. The error measure is the RMS torque error during the movement. The highest curve shows the performance of a nearest neighbor method without a local model. The nearest point was used to generate the torques for the feedforward command, which were then summed with the output from the feedback controller. The second

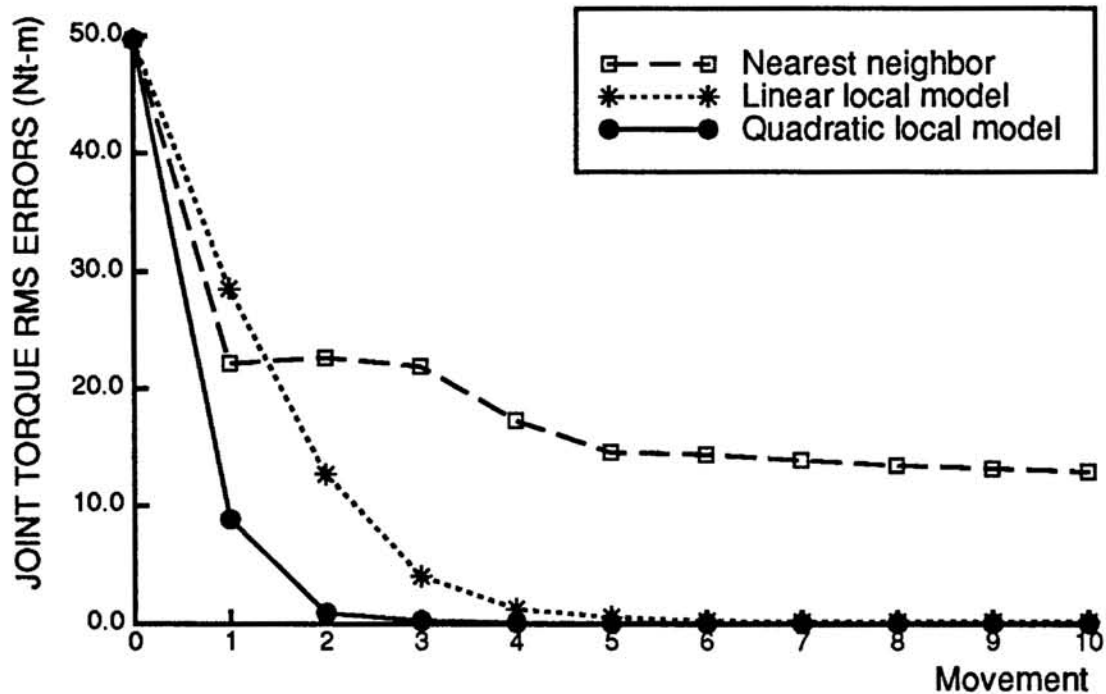

**Figure 2:** Learning curves from 3 different network designs on the two joint arm trajectory learning problem.

curve shows the performance using a linear local model. The third curve shows the performance using a quadratic local model. Adding the local model network greatly speeds up learning. The network with the quadratic local model learned more quickly than the one with the local linear model.

## 4   WHY DOES IT WORK?

In this learning paradigm the feedback controller serves as the teacher, or source of new data for the network. If the feedback controller is of poor quality, the nearest neighbor function approximation method tends to get "stuck" with a non-zero error level. The use of a local model seems to eliminate this stuck state, and reduce the dependence on the quality of the feedback controller.

Fast training is achieved by modularizing the network: the memory network does not need to search for weights in order to store the samples, and local models can be linear in the unknown parameters, leading to a single optimum which can be found by linear regression or gradient descent.

The combination of storing all the data and only using a certain number of nearby samples to form a local model minimizes interference between old and new data, and allows the range of generalization to depend on the density of the samples.

There are many issues left to explore. A disadvantage of this approach is the limited capacity of the memory network. In this version of the proposed neural network

architecture, every experience is stored. Eventually all the memory units will be used up. To use memory units more sparingly, only the experiences which are sufficiently different from previous experiences could be stored. Memory requirements could also be reduced by "forgetting" certain experiences, perhaps those that have not been referenced for a long time, or a randomly selected experience. It is an empirical question as to how large a memory capacity is necessary for this network design to be useful.

How should the distance metric be chosen? So far distance metrics have been devised by hand. Better distance metrics may be based on the stored data and a particular query. How far will this approach take us? Experiments using more complex systems and actual physical implementations, with the inevitable noise and high order dynamics, need to be done.

## Acknowledgments

B. Widrow and J. D. Cowan made the author aware of the work of Steinbuch and Taylor (Steinbuch and Widrow 1965, Cowan and Sharp 1988).

This paper describes research done at the Whitaker College, Department of Brain and Cognitive Sciences, Center for Biological Information Processing and the Artificial Intelligence Laboratory of the Massachusetts Institute of Technology. Support was provided under Office of Naval Research contract N00014-88-K-0321 and under Air Force Office of Scientific Research grant AFOSR-89-0500. Support for CGA was provided by a National Science Foundation Engineering Initiation Award and Presidential Young Investigator Award, an Alfred P. Sloan Research Fellowship, the W. M. Keck Foundation Assistant Professorship in Biomedical Engineering, and a Whitaker Health Sciences Fund MIT Faculty Research Grant.

## References

Cheng, P.E. (1984), "Strong Consistency of Nearest Neighbor Regression Function Estimators", *Journal of Multivariate Analysis,* 15:63-72.

Cleveland, W.S. (1979), "Robust Locally Weighted Regression and Smoothing Scatterplots", *Journal of the American Statistical Association,* 74:829-836.

Cleveland, W.S. and S.J. Devlin (1988), "Locally Weighted Regression: An Approach to Regression Analysis by Local Fitting", *Journal of the American Statistical Association,* 83:596-610.

Cleveland, W.S., S.J. Devlin and E. Grosse (1988), "Regression by Local Fitting: Methods, Properties, and Computational Algorithms", *Journal of Econometrics,* 37:87-114.

Cowan, J.D. and D.H. Sharp (1988), "Neural Nets", *Quarterly Reviews of Biophysics,* 21(3):365-427.

Crain, I.K. and B.K. Bhattacharyya (1967), "Treatment of nonequispaced two dimensional data with a digital computer", *Geoexploration,* 5:173-194.

Devroye, L.P. (1981), "On the Almost Everywhere Convergence of Nonparametric Regression Function Estimates", The Annals of Statistics, 9(6):1310-1319.

Eubank, R.L. (1988), *Spline Smoothing and Nonparametric Regression*, Marcel Dekker, New York, pp. 384-387.

Falconer, K.J. (1971), "A general purpose algorithm for contouring over scattered data points", Nat. Phys. Lab. Report NAC 6.

Farmer, J.D., and J.J. Sidorowich (1988), "Predicting Chaotic Dynamics", in *Dynamic Patterns in Complex Systems*, J.A.S. Kelso, A.J. Mandell, and M.F. Shlesinger, (eds.), World Scientific, New Jersey, pp. 265-292.

Farwig, R. (1987), "Multivariate Interpolation of Scattered Data by Moving Least Squares Methods", in J.C. Mason and M.G. Cox (eds), *Algorithms for Approximation*, Clarendon Press, Oxford, pp. 193-211.

Franke, R. (1982), "Scattered Data Interpolation: Tests of Some Methods", *Mathematics of Computation*, 38(157):181-200.

Franke, R. and G. Nielson (1980), "Smooth Interpolation of Large Sets of Scattered Data", *International Journal Numerical Methods Engineering*, 15:1691-1704.

Friedman, J.H., J.L. Bentley, and R.A. Finkel (1977), "An Algorithm for Finding Best Matches in Logarithmic Expected Time", *ACM Trans. on Mathematical Software*, 3(3):209-226.

Grosse, E. (1989), "LOESS: Multivariate Smoothing by Moving Least Squares", in C.K. Chui, L.L. Schumaker, and J.D. Ward (eds.), *Approximation Theory VI*, Academic Press, Boston, pp. 1-4.

Hillis, D. (1985), *The Connection Machine*, MIT Press, Cambridge, Mass.

Kazmierczak, H. and K. Steinbuch (1963), "Adaptive Systems in Pattern Recognition", *IEEE Transactions on Electronic Computers*, EC-12:822-835.

Lancaster, P. and K. Šalkauskas (1986), *Curve And Surface Fitting*, Academic Press, New York.

Legg, M.P.C. and R.P. Brent (1969), "Automatic Contouring", *Proc. 4th Australian Computer Conference*, 467-468.

Li, K.C. (1984), "Consistency for Cross-Validated Nearest Neighbor Estimates in Nonparametric Regression", *The Annals of Statistics*, 12:230-240.

Lodwick, G.D., and J. Whittle (1970), "A technique for automatic contouring field survey data", *Australian Computer Journal*, 2:104-109.

Macauley, F.R. (1931), *The Smoothing of Time Series*, National Bureau of Economic Research, New York.

McIntyre, D.B., D.D. Pollard, and R. Smith (1968), "Computer Programs For Automatic Contouring", *Kansas Geological Survey Computer Contributions 23*,

University of Kansas, Lawrence, Kansas.

**McLain, D.H.** (1974), "Drawing Contours From Arbitrary Data Points", *The Computer Journal*, 17(4):318-324.

**Müller, H.G.** (1987), "Weighted Local Regression and Kernel Methods for Nonparametric Curve Fitting", *Journal of the American Statistical Association*, 82:231-238.

**Palmer, J.A.B.** (1969), "Automated mapping", *Proc. 4th Australian Computer Conference*, 463-466.

**Pelto, C.R., T.A. Elkins, and H.A. Boyd** (1968), "Automatic contouring of irregularly spaced data", *Geophysics*, 33:424-430.

**Sheppard, W.F.** (1912), "Reduction of Errors by Means of Negligible Differences", *Proceedings of the Fifth International Congress of Mathematicians*, E. W. Hobson and A. E. H. Love (eds), Cambridge University Press, II:348-384.

**Sherriff, C.W.M.** (1920), "On a Class of Graduation Formulae", *Proceedings of the Royal Society of Edinburgh*, XL:112-128.

**Steinbuch, K.** (1961), "Die lernmatrix", *Kybernetik*, 1:36-45.

**Steinbuch, K. and U.A.W. Piske** (1963), "Learning Matrices and Their Applications", *IEEE Transactions on Electronic Computers*, EC-12:846-862.

**Steinbuch, K. and B. Widrow** (1965), "A Critical Comparison of Two Kinds of Adaptive Classification Networks", *IEEE Transactions on Electronic Computers*, EC-14:737-740.

**Stone, C.J.** (1975), "Nearest Neighbor Estimators of a Nonlinear Regression Function", *Proc. of Computer Science and Statistics: 8th Annual Symposium on the Interface*, pp. 413-418.

**Stone, C.J.** (1977), "Consistent Nonparametric Regression", *The Annals of Statistics*, 5:595-645.

**Stone, C.J.** (1982), "Optimal Global Rates of Convergence for Nonparametric Regression", *The Annals of Statistics*, 10(4):1040-1053.

**Taylor, W.K.** (1959), "Pattern Recognition By Means Of Automatic Analogue Apparatus", *Proceedings of The Institution of Electrical Engineers*, 106B:198-209.

**Taylor, W.K.** (1960), "A parallel analogue reading machine", *Control*, 3:95-99.

**Taylor, W.K.** (1964), "Cortico-thalamic organization and memory", *Proc. Royal Society B*, 159:466-478.

**Walters, R.F.** (1969), "Contouring by Machine: A User's Guide", *American Association of Petroleum Geologists Bulletin*, 53(11):2324-2340.

**Whittaker, E., and G. Robinson** (1924), *The Calculus of Observations*, Blackie & Son, London.